# Supervised graph inference

**Jean-Philippe Vert**
Centre de Géostatistique
Ecole des Mines de Paris
35 rue Saint-Honoré
77300 Fontainebleau, France
Jean-Philippe.Vert@mines.org

**Yoshihiro Yamanishi**
Bioinformatics Center
Institute for Chemical Research
Kyoto University
Uji, Kyoto 611-0011, Japan
yoshi@kuicr.kyoto-u.ac.jp

## Abstract

We formulate the problem of graph inference where part of the graph is known as a supervised learning problem, and propose an algorithm to solve it. The method involves the learning of a mapping of the vertices to a Euclidean space where the graph is easy to infer, and can be formulated as an optimization problem in a reproducing kernel Hilbert space. We report encouraging results on the problem of metabolic network reconstruction from genomic data.

## 1  Introduction

The problem of graph inference, or graph reconstruction, is to predict the presence or absence of edges between a set of points known to form the vertices of a graph, the prediction being based on observations about the points. This problem has recently drawn a lot of attention in computational biology, where the reconstruction of various biological networks, such as gene or molecular networks from genomic data, is a core prerequisite to the recent field of systems biology that aims at investigating the structures and properties of such networks. As an example, the *in silico* reconstruction of protein interaction networks [1], gene regulatory networks [2] or metabolic networks [3] from large-scale data generated by high-throughput technologies, including genome sequencing or microarrays, is one of the main challenges of current systems biology.

Various approaches have been proposed to solve the network inference problem. Bayesian [2] or Petri networks [4] are popular frameworks to model the gene regulatory or the metabolic network, and include methods to infer the network from data such as gene expression of metabolite concentrations [2]. In other cases, such as inferring protein interactions from gene sequences or gene expression, these models are less relevant and more direct approaches involving the prediction of edges between "similar" nodes have been tested [5, 6].

These approaches are unsupervised, in the sense that they base their prediction on prior knowledge about which edges should be present for a given set of points; this prior knowledge might for example be based on a model of conditional independence in the case of Bayesian networks, or on the assumption that edges should connect similar points. The actual situations we are confronted with, however, can often be expressed in a supervised framework: besides the data about the vertices, part of the network is already known. This

is obviously the case with all network examples discussed above, and the real challenge is to denoise the observed subgraph, if errors are assumed to be present, and to infer new edges involving in particular nodes outside of the observed subgraph. In order to clarify this point, let us take the example of an actual network inference problem that we treat in the experiment below: the inference of the metabolic network from various genomic data. The metabolic network is a graph of genes that involves only a subset of all the genes of an organisms, known as enzymes. Enzymes can catalyze chemical reaction, and an edge between two enzymes indicates that they can catalyze two successive reactions. For most organisms, this graph is partially known, because many enzymes have already been characterized. However many enzymes are also missing, and the problem is to detect uncharacterized enzymes and place them in their correct location in the metabolic network. Mathematically speaking, this means adding new edges involving new points, and eventually modifying edges in the known graph to remove mistakes from our current knowledge.

In this contribution we propose an algorithm for supervised graph inference, i.e., to infer a graph from observations about the vertices and from the knowledge of part of the graph. Several attempts have already been made to formalize the network inference problem as a supervised machine learning problem [1, 7], but these attempts consist in predicting each edge independently from each others using algorithms for supervised classification. We propose below a radically different setting, where the known subgraph is used to extract a new representation for the vertices, as points in a vector space, where the structure of the graph is easier to infer than from the original observations. The edge inference engine in the vector space is very simple (edges are inferred between nodes with similar representations), and the learning step is limited to the construction of the mapping of the nodes onto the vector space.

## 2   The supervised graph inference problem

Let us formally define the supervised graph inference problem. We suppose an undirected simple graph $G = (V, E)$ is given, where $V = (v_1, \ldots, v_n) \in \mathcal{V}^n$ is a set of vertices and $E \subset V \times V$ is a set of edges. The problem is, given an additional set of vertices $V' = (v'_1, \ldots, v'_m) \in \mathcal{V}^m$, to infer a set of edges $E' \subset V' \times (V \cup V') \cup (V \cup V') \times V'$ involving the nodes in $V'$. In many situations of interest, in particular gene networks, the additional nodes $V'$ might be known in advance, but we do not make this assumption here to ensure a level of generality as large as possible. For the applications we have in mind, the vertices can be represented in $\mathcal{V}$ by a variety of data types, including but not limited to biological sequences, molecular structures, expression profiles or metabolite concentrations. In order to allow this diversity and take advantage of recent works on positive definite kernels on general sets [8], we will assume that $\mathcal{V}$ is a set endowed with a positive definite kernel $k$, that is, a symmetric function $k : \mathcal{V}^2 \rightarrow \mathbb{R}$ satisfying $\sum_{i,j=1}^{p} a_i a_j k(x_i, x_j) \geq 0$ for any $p \in \mathbb{N}, (a_1, \ldots, a_n) \in \mathbb{R}^p$ and $(x_1, \ldots, x_p) \in \mathcal{V}^p$.

## 3   From distance learning to graph inference

Suppose first that a graph must be inferred on $p$ points $(x_1, \ldots, x_p)$ in the Euclidean space $\mathbb{R}^d$, without further information than "similar points" should be connected. Then the simplest strategy to predict edges between the points is to put an edge between vertices that are at a distance from each other smaller than a fixed threshold $\delta$. More or less edges can be inferred by varying the threshold. We call this strategy the "direct" strategy. We now propose to cast the supervised graph inference problem in a two step procedure:

- map the original points to a Euclidean space through a mapping $f : \mathcal{V} \rightarrow \mathbb{R}^d$;
- apply the direct strategy to infer the network on the points $\{f(v), v \in V \cup V'\}$.

While the second part of this procedure is fixed, the first part can be optimized by supervised learning of $f$ using the known network. To do so we require the mapping $f$ to map adjacent vertices in the known graph to nearby positions in $\mathbb{R}^d$, in order to ensure that the known graph can be recovered to some extent by the direct strategy. Stated this way, the problem of learning $f$ appears similar to a problem of distance learning that has been raised in the context of clustering [9], a important difference being that we need to define a new representation of the points and therefore a new (Euclidean) distance not only for the points in the training set, but also for points unknown during training.

Given a function $f : \mathcal{V} \to \mathbb{R}$, a possible criterion to assess whether connected (resp. disconnected) vertices are mapped onto similar (resp. dissimilar) points in $\mathbb{R}$ is the following:

$$R(f) = \frac{\sum_{(u,v)\in E} (f(u) - f(v))^2 - \sum_{(u,v)\notin E} (f(u) - f(v))^2}{\sum_{(u,v)\in V^2} (f(u) - f(v))^2}. \tag{1}$$

A small value of $R(f)$ ensures that connected vertices tend to be closer than disconnected vertices (in a quadratic error sense). Observe that the numerator ensures an invariance of $R(f)$ with respect to a scaling of $f$ by a constant, which is consistent with the fact that the direct strategy itself is invariant with respect to scaling of the points.

Let us denote by $f_V = (f(v_1), \ldots, f(v_n))^\top \in \mathbb{R}^n$ the values taken by $f$ on the training set, and by $L$ the combinatorial Laplacian of the graph $G$, i.e., the $n \times n$ matrix where $L_{i,j}$ is equal to $-1$ (resp. 0) if $i \neq j$ and vertices $v_i$ and $v_j$ are connected (resp. disconnected), and $L_{i,i} = -\sum_{j\neq i} L_{i,j}$. If we restrict $f_V$ to have zero mean ($\sum_{v\in V} f(v) = 0$), then the criterion (1) can be rewritten as follows:

$$R(f) = 4\frac{f_V^\top L f_V}{f_V^\top f_V} - 2.$$

The obvious minimum of $R(f)$ under the constraint $\sum_{v\in V} f(v) = 0$ is reached for any function $f$ such that $f_V$ is equal to the second largest eigenvector of $L$ (the largest eigenvector of $L$ begin the constant vector). However, this only defines the values of $f$ on the points $V$, but leaves indeterminacy on the values of $f$ outside of $V$. Moreover, any arbitrary choice of $f$ under a single constraint on $f_V$ is likely to be a mapping that overfits the known graph at the expense of the capacity to infer the unknown edges. To overcome both issues, we propose to regularize the criterion (1), by a smoothness functional on $f$, a classical approach in statistical learning [10, 11]. A convenient setting is to assume that $f$ belongs to the reproducing kernel Hilbert space (r.k.h.s.) $\mathcal{H}$ defined by the kernel $k$ on $\mathcal{V}$, and to use the norm of $f$ in the r.k.h.s. as a regularization operator. The regularized criterion to be minimized becomes:

$$\min_{f\in\mathcal{H}_0} \left\{ \frac{f_V^\top L f_V + \lambda\|f\|_{\mathcal{H}}^2}{f_V^\top f_V} \right\}, \tag{2}$$

where $\mathcal{H}_0 = \{f \in \mathcal{H} : \sum_{v\in V} f(v) = 0\}$ is the subset of $\mathcal{H}$ orthogonal to the function $x \mapsto \sum_{v\in V} k(x, v)$ in $\mathcal{H}$ and $\lambda$ is a regularization parameter.

We note that [12] have recently and indenpendently proposed a similar formulation in the context of clustering. The regularization parameter controls the trade-off between minimizing the original criterion (1) and ensuring that the solution has a small norm in the r.k.h.s. When $\lambda$ varies, the solution to (2) varies between to extremes:

- When $\lambda$ is small, $f_V$ tends to the second largest eigenvector of the Laplacian $L$. The regularization ensures that $f$ is well defined as a function of $\mathcal{V} \to \mathbb{R}$, but $f$ is likely to overfit the known graph.

- When $\lambda$ is large, the solution to (2) converges to the first kernel principal component (up to a scaling) [13], whatever the graph. Even though no supervised learning is performed in this case, one can observe that the resulting transformation, when the first $d$ kernel principal components are kept, is similar to the operation performed in spectral clustering [14, 15] where points are mapped onto the first few eigenvectors of a similarity matrix before being clustered.

Before showing how (2) is solved in practice, we must complete the picture by explaining how the mapping $f : \mathcal{V} \to \mathbb{R}^d$ is obtained. First note that the criterion in (2) is defined up to a scaling of the functions, and the solution is therefore a direction in the r.k.h.s. In order to extract a function, an additional constraint must be set, such that imposing the norm $||f||_{\mathcal{H}_V} = 1$, or imposing $\sum_{v \in V} f(v)^2 = 1$. The first solution correspond to an orthogonal projection onto the direction selected in the r.k.h.s. (which would for example give the same result as kernel PCA for large $\lambda$), while the second solution would provide a sphering of the data. We tested both possibilities in practice and found very little difference, with however slightly better results for the first solution (imposing $||f||_{\mathcal{H}_V} = 1$). Second, the problem (2) only defines a one-dimensional feature. In order to get a $d$-dimensional representation of the vertices, we propose to iterate the minimization of (2) under orthogonality constraints in the r.k.h.s., that is, we recursively define the $i$-th feature $f_i$ for $i = 1, \ldots, d$ by:

$$f_i = \operatorname*{arg\,min}_{f \in \mathcal{H}_0, f \perp \{f_1, \ldots, f_{i-1}\}} \left\{ \frac{f_V^\top L f_V + \lambda ||f||_{\mathcal{H}}^2}{f_V^\top f_V} \right\}. \tag{3}$$

## 4 Implementation

Let $k_V$ be the kernel obtained by centering $k$ on the set $V$, i.e.,

$$k_V(x, y) = k(x, y) - \frac{1}{n} \sum_{v \in V} k(x, v) - \frac{1}{n} \sum_{v \in V} k(y, v) + \frac{1}{n^2} \sum_{(v,v') \in V^2} k(v, v'),$$

and let $\mathcal{H}_V$ be the r.k.h.s. associated with $k_V$. Then it can easily be checked that $\mathcal{H}_V = \mathcal{H}_0$, where $\mathcal{H}_0$ is defined in the previous section as the subset of $\mathcal{H}$ of the function with zero mean on $V$. A simple extensions of the representer theorem [10] in the r.k.h.s. $\mathcal{H}_V$ shows that for any $i = 1, \ldots, d$, the solution to (3) has an expansion of the form:

$$f_i(x) = \sum_{j=1}^n \alpha_{i,j} k_V(x_j, x),$$

for some vector $\alpha_i = (\alpha_{i,1}, \ldots, \alpha_{i,n})^\top \in \mathbb{R}^n$. The corresponding vector $f_{i,V}$ can be written in terms of $\alpha_i$ by $f_{i,V} = K_V \alpha_i$, where $K_V$ is the Gram matrix of the kernel $k_V$ on the set $V$, i.e., $[K_V]_{i,j} = k_V(v_i, v_j)$ for $i, j = 1, \ldots, n$. $K_V$ is obtained from the Gram matrix $K$ of the original kernel $k$ by the classical formula $K_V = (I - U)K(I - U)$, $I$ being the $n \times n$ identity matrix and $U$ being the constant $n \times n$ matrix $[U]_{i,j} = 1/n$ for $i, j = 1, \ldots, n$ [13]. Besides, the norm in $\mathcal{H}_V$ is equal to $||f_i||_{\mathcal{H}_V}^2 = \alpha_i^\top K_V \alpha_i$, and the orthogonality constraint between $f_i$ and $f_j$ in $\mathcal{H}_V$ translates into $\alpha_i^\top K_V \alpha_j = 0$. As a result, the problem (2) is equivalent to the following:

$$\alpha_i = \operatorname*{arg\,min}_{\alpha \in \mathbb{R}^n, \alpha K_V \alpha_1 = \ldots = \alpha K_V \alpha_{i-1} = 0} \left\{ \frac{\alpha^\top K_V L K_V \alpha + \lambda \alpha^\top K_V \alpha}{\alpha^\top K_V^2 \alpha} \right\}. \tag{4}$$

Taking the differential of (4) with respect to $\alpha$ to 0 we see that the first vector $\alpha_1$ must solve the following generalized eigenvector problem with the smallest (non-negative) generalized eigenvalue:

$$(K_V L K_V + \lambda K_V) \alpha = \mu K_V^2 \alpha. \tag{5}$$

This shows that $\alpha_1$ must solve the following problem:

$$(LK_V + \lambda I)\,\alpha = \mu K_V \alpha, \qquad (6)$$

up to the addition of a vector $\epsilon$ satisfying $K\epsilon = 0$. Hence any solution of (5) differs from a solution of (6) by such an $\epsilon$, which however does not change the corresponding function $f \in \mathcal{H}_V$. It is therefore enough to solve (6) in order to find the first vector $\alpha_1$. $K$ being positive semidefinite, the other generalized eigenvectors of (6) are conjugate with respect to $K_V$, so it can easily be checked that the $d$ vectors $\alpha_1, \ldots, \alpha_d$ solving (4) are in fact the $d$ smallest generalized eigenvectors or (6). In practice, for large $n$, the generalized eigenvector problem (6) can be solved by first performing an incomplete Cholesky decomposition of $K_V$, see e.g. [16].

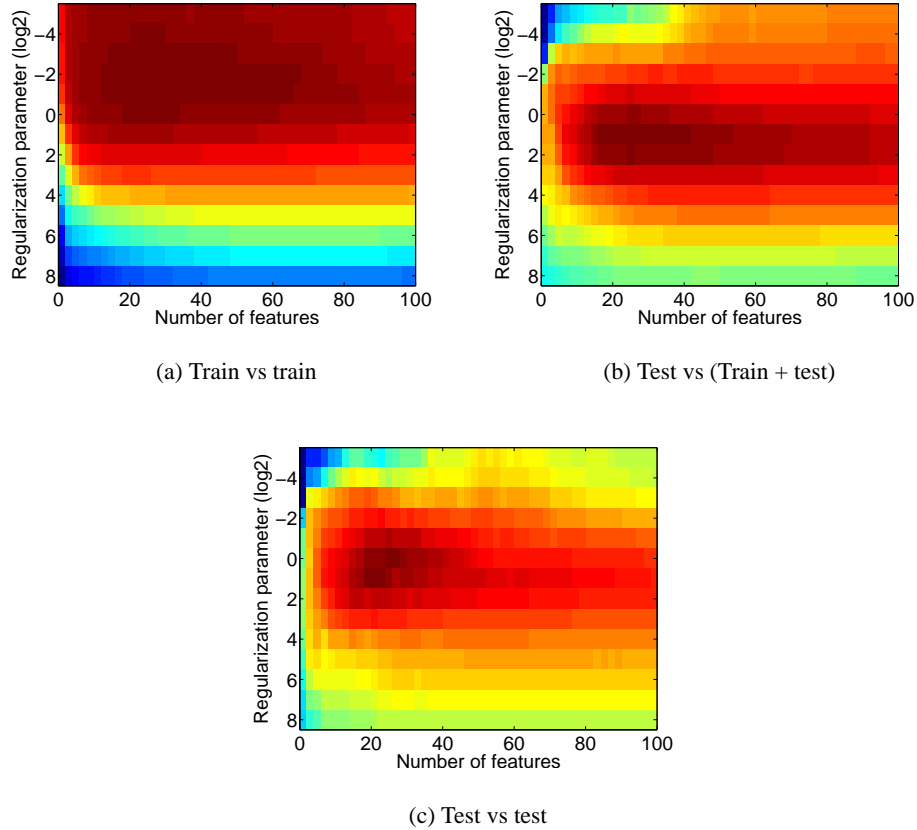

(a) Train vs train

(b) Test vs (Train + test)

(c) Test vs test

Figure 1: ROC score for different numbers of features and regularization parameters, in a 5-fold cross-validation experiment with the integrated kernel (the color scale is adjusted to highlight the variations inside each figure, the performance increases from blue to red).

## 5 Experiment

We tested the supervised graph inference method described in the previous section on the problem of inferring a gene network of interest in computational biology: the metabolic gene network, with enzymes present in an organism as vertices, and edges between two enzymes when they can catalyze successive chemical reactions [17]. Focusing on the budding

yeast *S. cerevisiae*, the network corresponding to our current knowledge of the network was extracted from the KEGG database [18]. The resulting network contains 769 vertices and 7404 edges. In order to infer it, various independent data about the genes can be used. We focus on three sources of data, likely to contain useful information to infer the graph: a set of 157 gene expression measurement obtained from DNA microarrays [19, 20], the phylogenetic profiles of the genes [21] as vectors of 145 bits indicating the presence or absence of each gene in 145 fully sequenced genomes, and their localization in the cell determined experimentally [22] as vectors of 23 bits indicating the presence of each gene into each of 23 compartment of the cell. In each case a Gaussian RBF kernel was used to represent the data as a kernel matrix. We denote these three datasets as "exp", "phy" and "loc" below. Additionally, we considered a fourth kernel obtained by summing the first three kernels. This is a simple approach to data integration that has proved to be useful in [23], for example. This integrated kernel is denoted "int" below.

We performed 5-fold cross-validation experiments as follows. For each random split of the set of genes into 80% (training set) and 20% (test set), the features are learned from the subgraph with genes from the training set as vertices. The edges involving genes in the test set are then predicted among all possible interactions involving the test set. The performance of the inference is estimated in term of ROC curves (the plot of the percentage of actual edges predicted as a function of the number of edges predicted although they are not present), and in terms of the area under the ROC curve normalized between 0 and 1. Notice that the set of possible interactions to be predicted is made of interactions between two genes in the test set, on the one hand, and between one gene in the test set and one gene in the training set, on the other hand. As it might be more challenging to infer an edge in the former case, we compute two performances: first on the edges involving two nodes in the test set, and second on the edges involving at least one vertex in the test set.

The algorithm contains 2 free parameters: the number $d$ of features to be kept, and the regularization parameter $\lambda$ that prevents from overfitting the known graph. We varied $\lambda$ among the values $2^i$, for $i = -5, \ldots, 8$, and $d$ between 1 and 100. Figure 1 displays the performance in terms of ROC index for the graph inference with the integrated kernel, for different values of $d$ and $\lambda$. On the training set, it can be seen that the effect of increasing $\lambda$ constantly decreases the performance of the graph reconstruction, which is natural since smaller values of $\lambda$ are expected to overfit the training graph. These results however justify that the criterion (1), although not directly related to the ROC index of the graph recon­struction procedure, is a useful criterion to be optimized. As an example, for very small values of $\lambda$, the ROC index on the training set is above 96%. The results on the test vs. test and on the test vs. (train + test) experiments show that overfitting indeed occurs for small $\lambda$ values, and that there is an optimum, both in terms of $d$ and $\lambda$. The slight difference between the performance landscapes in the experiments "test vs. test" and "test vs. (train + test)" show that the first one is indeed more difficult that the latter one, where some form of overfitting is likely to occur (in the mapping of the vertices in the training set). In par­ticular the "test vs. test" seems to be more sensitive to the number of features selected that the other setting. The absolute values of the ROC scures when 20 features are selected, for varying $\lambda$, are shown in figure 2. For all kernels tested, overfitting occurs at small $\lambda$ values, and an optimum exists (around $\lambda = 2 \sim 10$). The performance in the setting "test vs. (train+test)" is consistently better than that in the setting "test vs. test". Finally, and more interestingly, the inference with the integrated kernel outperforms the inference with each individual kernel. This is further highlighted in figure 3, where the ROC curves obtained for 20 features and $\lambda = 2$ are shown.

# References

[1] R. Jansen, H. Yu, D. Greenbaum, Y. Kluger, N.J. Krogan, S. Chung, A. Emili, M. Snyder, J.F. Greenblatt, and M. Gerstein. A bayesian networks approach for predicting protein-protein

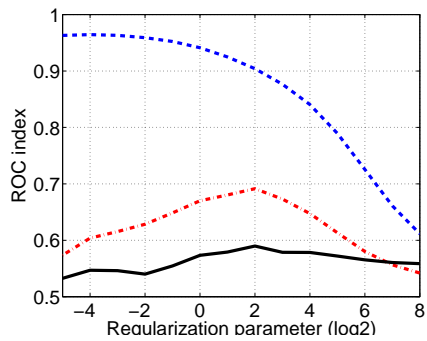

(a) Expression kernel

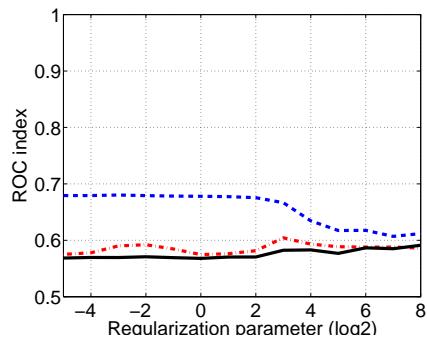

(b) Localization kernel)

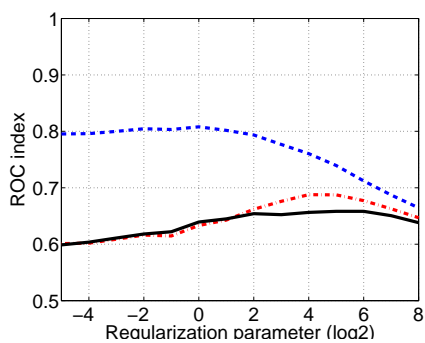

(c) Phylogenetic kernel

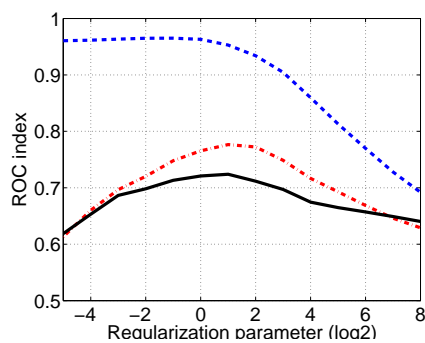

(d) Integrated kernel

Figure 2: ROC scores for different regularization parameters when 20 features are selected. Different pictures represent different kernels. In each picture, the dashed blue line, dash-dot red line and continuous black line correspond respectively to the ROC index on the training vs training set, the test vs (training + test) set, and the test vs test set.

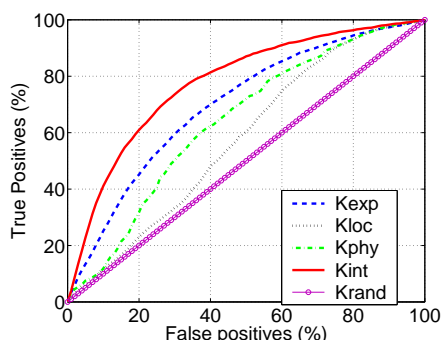

(a) Test vs. (train+test)

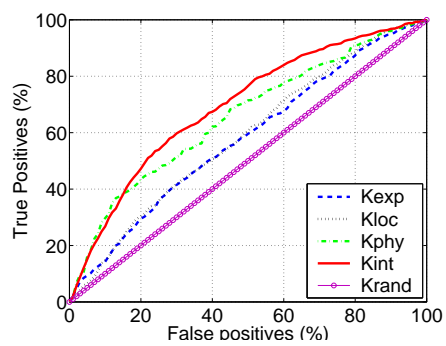

(b) Test vs. test

Figure 3: ROC with 20 features selected and $\lambda = 2$ for the various kernels.

interactions from genomic data. *Science*, 302(5644), 2003.

[2] N. Friedman, M. Linial, I. Nachman, and D. Pe'er. Using bayesian networks to analyze expression data. *Journal of Computational Biology*, 7:601–620, 2000.

[3] M. Kanehisa. Prediction of higher order functional networks from genomic data. *Pharmacogenomics*, 2(4):373–385, 2001.

[4] A. Doi, H. Matsuno, M. Nagasaki, and S. Miyano. Hybrid petri net representation of gene regulatory network. In *Proceedings of PSB 5*, pages 341–352, 2000.

[5] E.M. Marcotte, M. Pellegrini, H.-L. Ng, D.W. Rice, T.O. Yeates, and D. Eisenberg. Detecting protein function and protein-protein interactions from genome sequences. *Science*, 285(5428):751–753, 1999.

[6] F. Pazos and A. Valencia. Similarity of phylogenetic trees as indicator of protein?protein interaction. *Protein Engineering*, 9(14):609–614, 2001.

[7] J. R. Bock and D. A. Gough. Predicting protein-protein interactions from primary structure. *Bioinformatics*, 17:455–460, 2001.

[8] B. Schrölkopf, K. Tsuda, and J.-P. Vert. *Kernel methods in computational biology*. MIT Press, 2004.

[9] E.P. Xing, A.Y. Ng, M.I. Jordan, and S. Russell. Distance metric learning with application to clustering with side-information. In *NIPS 15*, pages 505–512. MIT Press, 2003.

[10] G. Wahba. *Splines Models for Observational Data*. Series in Applied Mathematics, Vol. 59, SIAM, Philadelphia, 1990.

[11] F. Girosi, M. Jones, and T. Poggio. Regularization theory and neural networks architectures. *Neural Computation*, 7(2):219–269, 1995.

[12] M. Belkin, P. Niyogi, and V. Sindhwani. Manifold regularization: A geometric framework for learning from examples. Technical Report TR-2004-06, University of Chicago, 2004.

[13] B. Schölkopf, A. J. Smola, and K.-R. Müller. Kernel principal component analysis. In B. Schölkopf, C. Burges, and A. Smola, editors, *Advances in Kernel Methods - Support Vector Learning*, pages 327–352. MIT Press, 1999.

[14] Y. Weiss. Segmentation using eigenvectors: a unifying view. In *Proceedings of the IEEE International Conference on Computer Vision*, pages 975–982. IEEE Computer Society, 1999.

[15] A. Y. Ng, M. I. Jordan, and Y. Weiss. On spectral clustering: Analysis and an algorithm. In *NIPS 14*, pages 849–856, MIT Press, 2002.

[16] F. R. Bach and M. I. Jordan. Kernel independent component analysis. *Journal of Machine Learning Research*, 3:1–48, 2002.

[17] J.-P. Vert and M. Kanehisa. Graph-driven features extraction from microarray data using diffusion kernels and kernel CCA. In *NIPS 15*. MIT Press, 2003.

[18] M. Kanehisa, S. Goto, S. Kawashima, and A. Nakaya. The KEGG databases at genomenet. *Nucleic Acids Research*, 30:42–46, 2002.

[19] P. T. Spellman, G. Sherlock, M. Q. Zhang, K. Anders, M. B. Eisen, P. O. Brown, D. Botstein, and B. Futcher. Comprehensive identification of cell cycle-regulated genes of the yeast *Saccharomyces cerevisiae* by microarray hybridization. *Mol. Biol. Cell*, 9:3273–3297, 1998.

[20] M. Eisen, P. Spellman, P. O. Brown, and D. Botstein. Cluster analysis and display of genome-wide expression patterns. *PNAS*, 95:14863–14868, 1998.

[21] M. Pellegrini, E. M. Marcotte, M. J. Thompson, D. Eisenberg, and T. O. Yeates. Assigning protein functions by comparative genome analysis: protein phylogenetic profiles. *PNAS*, 96(8):4285–4288, 1999.

[22] W.K. Huh, J.V. Falco, C. Gerke, A.S. Carroll, R.W. Howson, J.S. Weissman, and E.K. O'Shea. Global analysis of protein localization in budding yeast. *Nature*, 425:686–691, 2003.

[23] Y. Yamanishi, J.-P. Vert, A. Nakaya, and M. Kanehisa. Extraction of correlated gene clusters from multiple genomic data by generalized kernel canonical correlation analysis. *Bioinformatics*, 19:i323–i330, 2003.
